# A Probabilistic Model for Learning Concatenative Morphology

**Matthew G. Snover**
Department of Computer Science
Washington University
St Louis, MO, USA, 63130-4809
*ms9@cs.wustl.edu*

**Michael R. Brent**
Department of Computer Science
Washington University
St Louis, MO, USA, 63130-4809
*brent@cs.wustl.edu*

## Abstract

This paper describes a system for the unsupervised learning of morphological suffixes and stems from word lists. The system is composed of a generative probability model and hill-climbing and directed search algorithms. By extracting and examining morphologically rich subsets of an input lexicon, the directed search identifies highly productive paradigms. The hill-climbing algorithm then further maximizes the probability of the hypothesis. Quantitative results are shown by measuring the accuracy of the morphological relations identified. Experiments in English and Polish, as well as comparisons with another recent unsupervised morphology learning algorithm demonstrate the effectiveness of this technique.

## 1 Introduction

One of the fundamental problems in computational linguistics is adaptation of language processing systems to new languages with minimal reliance on human expertise. A ubiquitous component of language processing systems is the morphological analyzer, which determines the properties of morphologically complex words like *watches* and *gladly* by inferring their derivation as *watch+s* and *glad+ly*. The derivation reveals much about the word, such as the fact that *glad+ly* share syntactic properties with *quick+ly* and semantic properties with its stem *glad*. While morphological processes can take many forms, the most common are suffixation and prefixation (collectively, *concatenative morphology*).

In this paper, we present a system for unsupervised inference of morphological derivations of written words, with no prior knowledge of the language in question. Specifically, neither the stems nor the suffixes of the language are given in advance. This system is designed for concatenative morphology, and the experiments presented focus on suffixation. It is applicable to any language for written words lists are available. In languages that have been a focus of research in computational linguistics the practical applications are limited, but in languages like Polish, automated analysis of unannotated text corpora has potential applications for information retrieval and other language processing systems. In addition, automated analysis might find application as a hypothesis-generating tool for linguists or as a cognitive model of language acquisition. In this paper, however, we focus on the problem of unsupervised morphological inference for its inherent interest.

During the last decade several minimally supervised and unsupervised algorithms have been developed. Gaussier[1] describes an explicitly probabilistic system that is based primarily on spellings. It is an unsupervised algorithm, but requires the tweaking of parameters to tune it to the target language. Brent [2] and Brent et al. [3] describe Minimum Description Length, (MDL), systems. Goldsmith [4] describes a similar MDL approach. Our motivation in developing a new system was to improve performance and to have a model cast in an explicitly probabilistic framework. We are particularly interested in developing automated morphological analysis as a first stage of a larger grammatical inference system, and hence we favor a conservative analysis that identifies primarily productive morphological processes (those that can be applied to new words).

In this paper, we present a probabilistic model and search algorithm for automated analysis of suffixation, along with experiments comparing our system to that of Goldsmith [4]. This system, which extends the system of Snover and Brent [5], is designed to detect the final stem and suffix break of each word given a list of words. It does not distinguish between derivational and inflectional suffixation or between the notion of a stem and a root. Further, it does not currently have a mechanism to deal with multiple interpretations of a word, or to deal with morphological ambiguity. Within it's design limitations, however, it is both mathematically clean and effective.

## 2   Probability Model

This section introduces a prior probability distribution over the space of all hypotheses, where a hypothesis is a set of words, each with morphological split separating the stem and suffix. The distribution is based on a seven-step model for the generation of hypotheses, which is heavily based upon the probability model presented in [5]. The hypothesis is generated by choosing the number of stems and suffixes, the spellings of those stems and suffixes and then the combination of the stems and suffixes.

The seven steps are presented below, along with their probability distributions and a running example of how a hypothesis could be generated by this process. By taking the product over the distributions from all of the steps of the generative process, one can calculate the prior probability for any given hypothesis. What is described in this section is a mathematical model and not an algorithm intended to be run.

1. Choose the number of stems, $M$, according to the distribution:

$$\Pr(M) = \frac{6}{\pi^2} \left( \frac{1}{M} \right)^2 \tag{1}$$

   The $6/\pi^2$ term normalizes the inverse-squared distribution on the positive integers. The number of suffixes, $X$ is chosen according to the same probability distribution. The symbols M for steMs and X for suffiXes are used throughout this paper.
   *Example: M = 5. X = 3.*

2. For each stem $i$, choose its length in letters $L_i^m$, according to the inverse squared distribution. Assuming that the lengths are chosen independently and multiplying together their probabilities we have:

$$\Pr(L^m \mid M) = \left( \frac{6}{\pi^2} \right)^M \prod_{i=1}^{M} \left( \frac{1}{L_i^m} \right)^2 \tag{2}$$

   The distribution for the lengths of the suffixes, $L^x$, is similar to (2), differing only in that suffixes of length 0 are allowed, by offsetting the length by one.
   *Example: $L^m$ = 4, 4, 4, 3, 3. $L^x$ = 2, 0, 1.*

3. Let $\Sigma$ be the alphabet, and let $\{p_1 \ldots p_{|\Sigma|}\}$ be a probability distribution on $\Sigma$. For each $i$ from 1 to $M$, generate stem $i$ by choosing $L_i^m$ letters at random, according to the probabilities $\{p_1 \ldots p_{|\Sigma|}\}$. Call the resulting stem set STEM. The suffix set SUFF is generated in the same manner. The probability of any character, $l$, being chosen is obtained from a maximum likelihood estimate: $\hat{p}_l = \frac{c_l}{S}$ where $c_l$ is the count of $l$ among all the hypothesized stems and suffixes and $S = \sum_l c_l$.

The joint probability of the hypothesized stem and suffix sets is defined by the distribution:

$$\Pr(\text{STEM}, \text{SUFF} \mid M, L^m, X, L^x) = M! X! \prod_{l \in \Sigma} \left(\frac{c_l}{S}\right)^{c_l} \qquad (3)$$

The factorial terms reflect the fact that the stems and suffixes could be generated in any order.

*Example: STEM = {walk, look, door, far, cat}. SUFF = {ed, $\epsilon$, s}.*

4. We now choose the number of paradigms, $P$. A paradigm is a set of suffixes and the stems that attach to those suffixes and no others. Each stem is in exactly one paradigm, and each paradigm has at least one stem., thus $P$ can range from 1 to $M$. We pick $P$ according to the following uniform distribution:

$$\Pr(P \mid M) = M^{-1} \qquad (4)$$

*Example: P = 3.*

5. We choose the number of suffixes in the paradigms, $D$, according to a uniform distribution. The distribution for picking $D_i$, suffixes for paradigm $i$ is:

$$\Pr(D_i \mid XP) = \frac{1}{X}$$

The joint probability over all paradigms, $D$ is therefore:

$$\Pr(D \mid XP) = \prod_{i=1}^{P} X^{-1} = \left(\frac{1}{X}\right)^P \qquad (5)$$

*Example: D = {2, 1, 2}.*

6. For each paradigm $i$, choose the set of $D_i$ suffixes, $\text{PARA}_i^x$ that the paradigm will represent. The number of subsets of a given size is finite so we can again use the uniform distribution. This implies that the probability of each individual subset of size $D_i$, is the inverse of the total number of such subsets. Assuming that the choices for each paradigm are independent:

$$\Pr(\text{PARA}^x \mid XPD) = \prod_{i=1}^{P} \binom{X}{D_i}^{-1} = \binom{X}{D_i}^{-P} \qquad (6)$$

*Example: PARA$_1^x$ = {$\epsilon$, s, ed}. PARA$_2^x$ = {$\epsilon$}. PARA$_3^x$ = {$\epsilon$, s}.*

7. For each stem choose the paradigm that the stem will belong in, according to a distribution that favors paradigms with more stems. The probability of choosing a paradigm $i$, for a stem is calculated using a maximum likelihood estimate:

$$\frac{|\text{PARA}_i^m|}{M}$$

where $\text{PARA}_i^m$ is the set of stems in paradigm $i$. Assuming that all these choices are made independently yields the following:

$$\Pr(\text{PARA}^m \mid MXP) = \prod_{i=1}^{P} \left(\frac{|\text{PARA}_i^m|}{M}\right)^{|\text{PARA}_i^m|} \qquad (7)$$

*Example: PARA$_1^m$ = {walk, look}. PARA$_2^m$ = {far}. PARA$_3^m$ = {door, cat}.*

Combining the results of stages 6 and 7, one can see that the running example would yield the hypothesis consisting of the set of words with suffix breaks, {walk+$\epsilon$, walk+s, walk+ed, look+$\epsilon$, look+s, look+ed, far+$\epsilon$, door+$\epsilon$, door+s, cat+$\epsilon$, cat+s}. Removing the breaks in the words results in the set of input words. To find the probability for this hypothesis just take of the product of the probabilities from equations (1) to (7).

Using this generative model, we can assign a probability to any hypothesis. Typically one wishes to know the probability of the hypothesis given the data, however in our case such a distribution is not required. Equation (8) shows how the probability of the hypothesis given the data could be derived from Bayes law.

$$\Pr(\text{Hyp} \mid \text{Data}) = \frac{\Pr(\text{Hyp}) \Pr(\text{Data} \mid \text{Hyp})}{\Pr(\text{Data})} \tag{8}$$

Our search only considers hypotheses consistent with the data. The probability of the data given the hypothesis, $\Pr(\text{Data}|\text{Hyp})$, is always 1, since if you remove the breaks from any hypothesis, the input data is produced. This would not be the case if our search considered inconsistent hypotheses. The prior probability of the data is constant over all hypotheses, thus the probability of the hypothesis given the data reduces to $\Pr(\text{Hyp})/c$. The prior probability of the hypothesis is given by the above generative process and, among all consistent hypotheses, the one with the greatest prior probability also has the greatest posterior probability.

# 3 Search

This section details a novel search algorithm which is used to find a high probability segmentation of the all the words in the input lexicon, $L$. The input lexicon is a list of words extracted from a corpus. The output of the search is a segmentation of each of the input words into a stem and suffix.

The search algorithm has two phases, which we call the *directed* search and the *hill-climbing* search. The directed search builds up a consistent hypothesis about the segmentation of all words in the input out of consistent hypothesis about subsets of the words. The hill-climbing search further tunes the result of the directed search by trying out nearby hypotheses over all the input words.

## 3.1 Directed Search

The directed search is accomplished in two steps. First, sub-hypotheses, each of which is a hypothesis about a subset of the lexicon, are examined and ranked. The $N$ best sub-hypotheses are then incrementally combined until a single sub-hypothesis remains. The remainder of the input lexicon is added to this sub-hypothesis at which point it becomes the final hypothesis.

We define the set of possible suffixes to be the set of terminal substrings, including the empty string $\epsilon$, of the words in $L$. For each subset of the possible suffixes $X$, there is a maximal set of possible stems (initial substrings) $M_X$, such that for each $x \in X$ and each $m \in M_X$, $mx$ is a word in $L$. We define $h(X)$ to be the sub-hypothesis in which each input word $mx$ that can be analyzed as consisting of a stem in $M_X$ and a suffix in $X$ is analyzed that way. This subhypothesis consists of all pairings of the stems in $M_X$ and the suffixes in $X$ with the corresponding morphological breaks. One can think of each sub-hypothesis as initially corresponding to a maximally filled paradigm. We only consider sub-hypotheses which have at least two stems and two suffixes.

For each sub-hypothesis, $h$, there is a corresponding null hypothesis, $\bar{h}$, which has the same set of words as $h$, but in which all the words are hypothesized to consist of the

word as the stem and $\epsilon$ as the suffix. We give each sub-hypothesis a score as follows: $\text{score}(h) = \Pr(h)/\Pr(\bar{h})$. This reflects how much more probable $h$ is for those words, than the null hypothesis.

One can view all sub-hypotheses as nodes in a directed graph. Each node, $n_i$, is connected to another node, $n_j$ if and only if $n_j$ represents a superset of the suffixes that $n_i$ represents, which is exactly one suffix greater in size than the set that $n_i$ represents. By beginning at the node representing no suffixes, one can apply standard graph search techniques, such as a beam search or a best first search to find the $N$ best scoring nodes without visiting all nodes. While one cannot guarantee that such approaches perform exactly the same as examining all sub-hypotheses, initial experiments using a beam search with a beam size equal to $N$, with a $N$ of 100, show that the $N$ best sub-hypotheses are found with a significant decrease in the number of nodes visited. The experiments presented in this paper do not use these pruning methods.

The $N$ highest scoring sub-hypotheses are incrementally combined in order to create a hypothesis over the complete set of input words. Changing the value of $N$ does not dramatically alter the results of the algorithm, though higher values of $N$ give slightly better results. We let $N$ be 100 in the experiments reported here.

Let $S$ be the $N$ highest scoring sub-hypotheses. We iteratively remove the highest scoring hypothesis $s'$ from $S$. The words in $s'$ are added to each of the remaining sub-hypotheses in $S$, and their null hypotheses, with their morphological breaks from $s'$. If a word in $s'$ was already in $s$ the morphological break from $s'$ overrides the one from $s$. All of the sub-hypotheses are now rescored, as the words in them have changed. If, after rescoring, none of the sub-hypotheses have likelihood ratios greater than one, then we use $s'$ as our final hypothesis. Otherwise we, iterate until either there is only one sub-hypotheses left or all subhypotheses have scores no greater than one.

The final sub-hypothesis, $s'$, is now converted into a full hypothesis over all the words. All words in $L$ that are not in $s'$ are added to $s'$ with suffix $\epsilon$.

### 3.2 Hill Climbing Search

The hill climbing search further optimizes the probability of the hypothesis by moving stems to new nodes. For each possible suffix $x$, and each node $n$, the search attempts to add $x$ to $n$. This means that all stems in $n$ that can take the suffix $x$ are moved to a new node, $n'$, which represents all the suffixes of $n$ and $x$. This is analogous to pushing stems to adjacent nodes in a directed graph. A stem $m$, can only be moved into a node with the suffix $x$, if the new word, $mx$ is an observed word in the input lexicon. The move is only done if it increases the probability of the hypothesis.

There is an analogous suffix removal step which attempts to remove suffixes from nodes. The hill climbing search continues to add and remove suffixes to nodes until the probability of the hypothesis cannot be increased. A more detailed description of this portion of the search and its algorithmic invariants is given in [5].

## 4 Experiment and Evaluation

### 4.1 Experiment

We tested our unsupervised morphology learning system, which we refer to as Paramorph, and Goldsmith's MDL system, otherwise known as Linguistica[1], on various sized word lists

from English and Polish corpora. For English we used set A of the Hansard corpus, which is a parallel English and French corpus of proceedings of the Canadian Parliament. We were unable to find a standard corpus for Polish and developed one from online sources. The sources for the Polish corpus were older texts and thus our results correspond to a slightly antiquated form of the language. The results were evaluated by measuring the accuracy of the stem relations identified.

We extracted input lexicons from each corpus, excluding words containing non-alphabetic characters. The 100 most common words in each corpus were also excluded, since these words tend to be function words and are not very informative for morphology. The systems were run on the 500, 1,000, 2,000, 4,000, and 8,000 most common remaining words. The experiments in English were also conducted on the 16,000 most common words from the Hansard corpus.

### 4.1.1 Stem Relation

Ideally, we would like to be able to specify the correct morphological break for each of the words in the input, however morphology is laced with ambiguity, and we believe this to be an inappropriate method for this task. For example it is unclear where the break in the word, "location" should be placed. It seems that the stem "locate" is combined with the suffix "tion", but in terms of simple concatenation it is unclear if the break should be placed before or after the "t".

In an attempt to solve this problem we have developed a new measure of performance, which does not specify the exact morphological split of a word. We measure the accuracy of the stems predicted by examining whether two words which are morphologically related are predicted as having the same stem. The actual break point for the stems is not evaluated, only whether the words are predicted as having the same stem. We are working on a similar measure for suffix identification.

Two words are related if they share the same immediate stem. For example the words "building", "build", and "builds" are related since they all have "build" as a stem, just as "building" and "buildings" are related as they both have "building" as a stem. The two words, "buildings" and "build" are not directly related since the former has "building" as a stem, while "build" is its own stem. Irregular forms of words are also considered to be related even though such relations would be very difficult to detect with a simple concatenation model.

The stem relation precision measures how many of the relations predicted by the system were correct, while the recall measures how many of the relations present in the data were found. Stem relation fscore is an unbiased combination of precision and recall that favors equal scores.

### 4.2 Results

The results from the experiments are shown in Figures 1 and 2. All graphs are shown using a log scale for the corpus size. Due to software difficulties we were unable to get Linguistica to run on 500, 1000, and 2000 words in English. The software ran without difficulties on the larger English datasets and on the Polish data. As an additional note, Linguistica was dramatically faster than Paramorph, which is a development oriented software package and not as optimized for efficient runtime as Linguistica appears to be.

Figure 1 shows the number of different suffixes predicted by each of the algorithms in both English and Polish. Our Paramorph system found a relatively constant number of

other parameters were left at their default values.

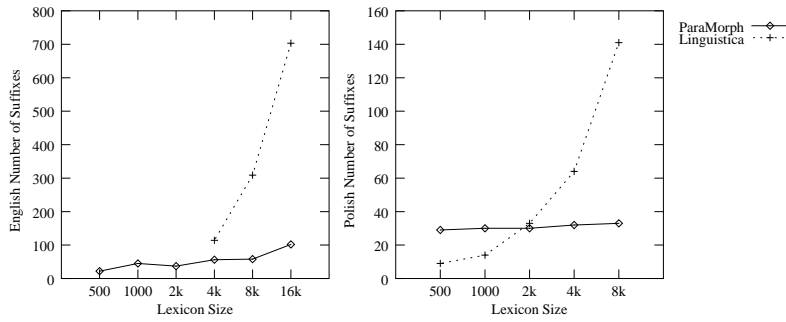

Figure 1: Number of Suffixes Predicted

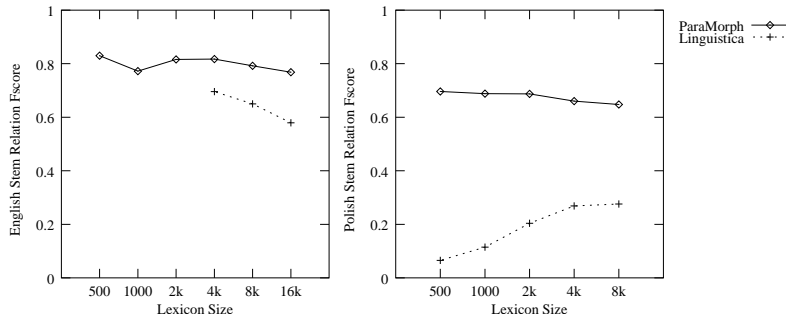

Figure 2: Stem Relation Fscores

suffixes across lexicon sizes and Linguistica found an increasingly large number of suffixes, predicting over 700 different suffixes in the 16,000 word English lexicon.

Figure 2 shows the fscores using the stem relation metric for various sizes of English and Polish input lexicons. Paramorph maintains a very high precision across lexicon sizes in both languages, whereas the precision of Linguistica decreases considerably at larger lexicon sizes. However Linguistica shows an increasing recall as the lexicon size increases, with Paramorph having a decreasing recall as lexicon size increases, though the recall of Linguistica in Polish is consistently lower than the Paramorph's recall. The fscores for Paramorph and Linguistica in English are very close, and Paramorph appears to clearly outperform Linguistica in Polish.

| Suffixes | Stems |
|---|---|
| -a -e -ego -ej -ie -o -y | dziwn |
| $\epsilon$ -a -ami -y -ę | chmur siekier |
| $\epsilon$ -cie -li -m -ć | gada odda sprzeda |

Table 1: Sample Paradigms in Polish

Table 1 shows several of the larger paradigms found by Paramorph when run on 8000 words of Polish. The first paradigm shown is for the single adjective stem meaning "strange" with numerous inflections for gender, number and case, as well as one derivational suffix, "-ie" which changes it into an adverb, "strangely". The second paradigm is for the nouns, "cloud" and "ax", with various case inflections and the third paradigm paradigm contains the verbs, "talk", "return", and "sell". All suffixes in the third paradigm are inflectional indicating tense and agreement.

The differences between the performance of Linguistica and Paramorph can most easily be seen in the number of suffixes predicted by each algorithm. The number of suffixes predicted by Linguistica grows linearly with the number of words, in general causing his algorithm to get much higher recall at the expense of precision. Paramorph maintains a fairly constant number of suffixes, causing it to generally have higher precision at the expense of recall. This is consistent with our goals to create a conservative system for morphological analysis, where the number of false positives is minimized.

The Polish language presents special difficulties for both Linguistica and Paramorph, due to the highly complex nature of its morphology. There are far fewer spelling change rules and a much higher frequency of suffixes in Polish than in English. In addition phonology plays a much stronger role in Polish morphology, causing alterations in stems, which are difficult to detect using a concatenative framework.

## 5 Discussion

Many of the stem relations predicted by Paramorph result from postulating stem and suffix breaks in words that are actually morphologically simple. This occurs when the endings of these words resemble other, correct, suffixes. In an attempt to deal with this problem we have investigated incorporating semantic information into the probability model since morphologically related words also tend to be semantically related. A successful implementation of such information should eliminate errors such as *capable* breaking down as *cap+able* since *capable* is not semantically related to *cape* or *cap*.

The goal of the Paramorph system was to produce a preliminary description, with very low false positives, of the final suffixation, both inflectional and derivational, in a language independent manner. Paramorph performed better for the most part with respect to Fscore than Linguistica, but more importantly, the precision of Linguistica does not approach the precision of our algorithm, particularly on the larger corpus sizes. In summary, we feel our Paramorph system has attained the goal of producing an initial estimate of suffixation that could serve as a front end to aid other models in discovering higher level structure.

**References**

[1] Éric. Gaussier. 1999. Unsupervised learning of derivational morphology from inflectional lexicons. In *ACL '99 Workshop Proceedings: Unsupervised Learning in Natural Language Processing*. ACL.

[2] Michael R. Brent. 1993. Minimal generative models: A middle ground between neurons and triggers. In *Proceedings of the Fifth International Workshop on Artificial Intelligence and Statistics*, Ft. Laudersdale, FL.

[3] Michael R. Brent, Sreerama K. Murthy, and Andrew Lundberg. 1995. Discovering morphemic suffixes: A case study in minimum description length induction. In *Proceedings of the 15th Annual Conference of the Cognitive Science Society*, pages 28-36, Hillsdale, NJ. Erlbaum.

[4] John Goldsmith. 2001. Unsupervised learning of the morphology of a natural language. *Computational Linguistics*, 27:153-198.

[5] Matthew G. Snover and Michael R. Brent. 2001. A Bayesian Model for Morpheme and Paradigm Identification. In *Proceedings of the 39th Annual Meeting of the ACL*, pages 482-490. ACL.

## Footnotes

[1] A demo version available on the web, http://humanities.uchicago.edu/faculty/goldsmith/, was used for these experiments. Word-list corpus mode and the method A suffix detection were used. All
